# Structured sparse coding via lateral inhibition

**Karol Gregor**
Janelia Farm, HHMI
19700 Helix Drive
Ashburn, VA, 20147
karol.gregor@gmail.com

**Arthur Szlam**
The City College of New York
Convent Ave and 138th st
New York, NY, 10031
aszlam@courant.nyu.edu

**Yann LeCun**
New York University
715 Broadway, Floor 12
New York, NY, 10003
yann@cs.nyu.edu

## Abstract

This work describes a conceptually simple method for structured sparse coding and dictionary design. Supposing a dictionary with $K$ atoms, we introduce a structure as a set of penalties or interactions between every pair of atoms. We describe modifications of standard sparse coding algorithms for inference in this setting, and describe experiments showing that these algorithms are efficient. We show that interesting dictionaries can be learned for interactions that encode tree structures or locally connected structures. Finally, we show that our framework allows us to learn the values of the interactions from the data, rather than having them pre-specified.

## 1   Introduction

Sparse modeling (Olshausen and Field, 1996; Aharon et al., 2006) is one of the most successful recent signal processing paradigms. A set of $N$ data points $X$ in the Euclidean space $\mathbb{R}^d$ is written as the approximate product of a $d \times k$ dictionary $W$ and $k \times N$ coefficients $Z$, where each column of $Z$ is penalized for having many non-zero entries. In equations, if we take the approximation to $X$ in the least squares sense, and the penalty on the coefficient matrix to be the $l_1$ norm, we wish to find

$$\operatorname{argmin}_{Z,W} \sum_k ||Wz_k - x_k||^2 + \lambda||z_k||_1. \tag{1}$$

In (Olshausen and Field, 1996), this model is introduced as a possible explanation of the emergence of orientation selective cells in the primary visual cortex V1; the matrix representing $W$ corresponds to neural connections.

It is sometimes appropriate to enforce more structure on $Z$ than just sparsity. For example, we may wish to enforce a tree structure on $Z$, so that certain basis elements can be used by any data point, but others are specific to a few data points; or more generally, a graph structure on $Z$ that specifies which elements can be used with which others. Various forms of structured sparsity are explored in (Kavukcuoglu et al., 2009; Jenatton et al., 2010; Kim and Xing, 2010; Jacob et al., 2009; Baraniuk et al., 2009). From an engineering perspective, structured sparse models allow us to access or enforce information about the dependencies between codewords, and to control the expressive power of the model without losing reconstruction accuracy. From a biological perspective, structured sparsity is interesting because structure and sparsity are present in neocortical representations. For example, neurons in the same mini-columns of V1 are receptive to similar orientations and activate together. Similarly neurons within columns in the inferior temporal cortex activate together and correspond to object parts.

In this paper we introduce a new formulation of structured sparsity. The $l_1$ penalty is replaced with a set of interactions between the coding units corresponding to intralayer connections in the neocortex. For every pair of units there is an interaction weight that specifies the cost of simultaneously activating both units. We will describe several experiments with the model. In the first set of experiments

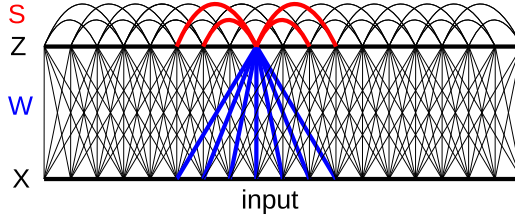

Figure 1: Model (3) in the locally connected setting of subsection 3.3. Code units are placed in two dimensional grid above the image (here represented in 1-$d$ for clarity). A given unit connects to a small neighborhood of an input via $W$ and to a small neighborhood of code units via $S$. The $S$ is present and positive (inhibitory) if the distance $d$ between units satisfies $r_1 < d < r_2$ for some radii.

we set the interactions to reflect a prespecified structure. In one example we create a locally connected network with inhibitory connections in a ring around every unit. Trained with natural images, this leads to dictionaries with Gabor-like edge elements with similar orientations placed in nearby locations, leading to the pinwheel patterns analogous to those observed in V1 of higher mammals. We also place the units on a tree and place inhibitory interactions between different branches of the tree, resulting in edges of similar orientation being placed in the same branch of the tree, see for example (Hyvarinen and Hoyer, 2001). In the second set of experiments we learn the values of the lateral connections instead of setting them, in effect learning the structure. When trained on images of faces, the system learns to place different facial features at correct locations in the image.

The rest of this paper is organized as follows: in the rest of this section, we will introduce our model, and describe its relationship between the other structured sparsity mentioned above. In section 2, we will describe the algorithms we use for optimizing the model. Finally, in section 3, we will display the results of experiments showing that the algorithms are efficient, and that we can effectively learn dictionaries with a given structure, and even learn the structure.

## 1.1 Structured sparse models

We start with a model that creates a representation $Z$ of data points $X$ via $W$ by specifying a set of disallowed index pairs of $Z$: $U = \{(i_1, j_1), (i_2, j_2), ..., (i_k j_k)\}$ - meaning that representations $Z$ are not allowed if both $Z_i \neq 0$ and $Z_j \neq 0$ for any given pair $(i, j) \in U$. Here we constrain $Z \geq 0$. The inference problem can be formulated as

$$\min_{Z \geq 0} \sum_{j=1}^{N} ||WZ_j - X_j||^2,$$

subject to

$$ZZ^T(i, j) = 0, \ i, j \in U.$$

Then the Langrangian of the energy with respect to $Z$ is

$$\sum_{j=1}^{N} ||WZ_j - X_j||^2 + Z_j^T S Z_j, \tag{2}$$

where $S_{ij}$ are the dual variables to each of the constraints in $U$, and are 0 in the unconstrained pairs. A local minimum of the constrained problem is a saddle point for (2). At such a point, $S_{ij}$ can be interpreted as the weight of the inhibitory connection between $W_i$ and $W_j$ necessary to keep them from simultaneously activating. This observation will be the starting point for this paper.

## 1.2 Lateral inhibition model

In practice, it is useful to soften the constraints in $U$ to a fixed, prespecified penalty, instead of a maximization over $S$ as would be suggested by the Lagrangian form. This allows some points to

use proscribed activations if they are especially important for the reconstruction. To use units with both positive and negative activations we take absolute values and obtain

$$\min_{W,Z} \sum_j ||WZ_j - X_j||^2 + |Z_j|^T S |Z_j|, \tag{3}$$

$$||W_j|| = 1 \ \forall j.$$

where $|Z_j|$ denotes the vector obtained from the vector $Z_j$ by taking absolute values of each component, and $Z_j$ is the $j$th column of $Z$. $S$ will usually be chosen to be symmetric and have 0 on the diagonal. As before, instead of taking absolute values, we can instead constrain $Z \geq 0$ allowing to write the penalty as $Z_j^T S Z_j$. Finally, note that we can also allow $S$ to be negative, implementing excitatory interaction between neurons. One then has to prevent the sparsity term to go to minus infinity by limiting the amount of excitation a given element can experience (see the algorithm section for details).

The Lagrangian optimization tries to increase the inhibition between a forbidden pair whenever it activates. If our goal is to learn the interactions, rather than enforce the ones we have chosen, then it makes sense to do the opposite, and decrease entries of $S$ corresponding to pairs which are often activated simultaneously. To force a nontrivial solution and encourage $S$ to economize a fixed amount of inhibitory power, we also propose the model

$$\min_S \min_{W,Z} \sum_j ||WZ_j - X_j||^2 + |Z_j|^T S |Z_j|, \tag{4}$$

$$Z \geq 0, \ ||W_j|| = 1 \ \forall j,$$

$$0 \leq S \leq \beta, \ S = S^T, \ \text{and} |S_j|_1 = \alpha \ \forall j$$

Here, $\alpha$ and $\beta$ control the total inhibitory power of the activation of an atom in $W$, and how much the inhibitory power can be concentrated in a few interactions (i.e. the sparsity of the interactions). As above, usually one would also fix $S$ to be 0 on the diagonal.

## 1.3   Lateral inhibition and weighted $l_1$

Suppose we have fixed $S$ and $W$, and are inferring $z$ from a datapoint $x$. Furthermore, suppose that a subset $I$ of the indices of $z$ do not inhibit each other. Then if $I^c$ is the complement of $I$, for any fixed value of $z_{I^c}$ (here the subscript refers to indices of the column vector $z$), the cost of using $z_I$ is given by

$$||W_I z_I - x||^2 + \sum_{i \in I} \lambda_i |z_i|,$$

where $\lambda_i = \sum_{j \in I^c} S_{ij} |z_j|$. Thus for $z_{I^c}$ fixed, we get a weighted lasso in $z_I$.

## 1.4   Relation with previous work

As mentioned above, there is a growing literature on structured dictionary learning and structured sparse coding. The works in (Baraniuk et al., 2009; Huang et al., 2009) use a greedy approach for structured sparse coding based on OMP or CoSaMP. These methods are fast when there is an efficient method for searching the allowable additions to the active set of coefficients at each greedy update, for example if the coefficients are constrained to lie on a tree. These works also have provable recovery properties when the true coefficients respect the structure, and when the dictionaries satisify certain incoherence properites. A second popular basic framework is group sparsity (Kavukcuoglu et al., 2009; Jenatton et al., 2010; Kim and Xing, 2010; Jacob et al., 2009). In these works the coefficients are arranged into a predetermined set of groups, and the sparsity term penalizes the number of active groups, rather than the number of active elements. This approach has the advantage that the resulting inference problems are convex, and many of the works can guarantee convergence of their inference schemes to the minimal energy.

In our framework, the interactions in $S$ can take any values, giving a different kind of flexibility. Although our framework does not have a convex inference, the algorithms we propose experimentally efficiently find good codes for every $S$ we have tried. Also note that in this setting, recovery theorems with incoherence assumptions are not applicable, because we will learn the dictionaries, and

so there is no guarantee that the dictionaries will satisfy such conditions. Finally, a major difference between the methods presented here and those in the other works is that we can learn the $S$ from the data simultaneously with dictionary; as far as we know, this is not possible via the above mentioned works.

The interaction between a set of units of the form $z^T R z + \theta^T z$ was originally used in Hopfield nets (Hopfield, 1982); there the $z$ are binary vectors and the inference is deterministic. Boltzman machines (Ackley et al., 1985) have a similar term, but the $z$ and the inference are stochastic, e.g. Markov chain Monte carlo. With $S$ fixed, one can consider our work a special case of real valued Hopfield nets with $R = W^T W + S$ and $\theta = W^T x$; because of the form of $R$ and $\theta$, fast inference schemes from sparse coding can be used. When we learn $S$, the constraints on $S$ serve the same purpose as the contrastive terms in the updates in a Boltzman machine.

In (Garrigues and Olshausen, 2008) lateral connections were modeled as the connections of an Ising model with the Ising units deciding which real valued units (from which input was reconstructed) were on. The system learned to typically connect similar orientations at a given location. Our model is related but different - it has no second layer, the lateral connections control real instead of binary values and the inference and learning is simpler, at the cost of a true generative model. In (Druckmann and Chklovskii, 2010) the lateral connections were trained so that solutions $z_t$ to a related ODE starting from the inferred code of $z = z_0$ of an input $x$ would map via $W$ to points close to $x$. In that work, the lateral connections were trained in response to the dictionary, rather than simultaneously with it, and did not participate in inference.

In (Garrigues and Olshausen, 2010) the coefficients were given by a Laplacian scale mixture prior, leading to multiplicative modulation, as in this work. However, in contrast, in our case the sparsity coefficients are modulated by the units in the same layer, and we learn the modulation, as opposed to the fixed topology in (Garrigues and Olshausen, 2010).

## 2   Algorithms

In this section we will describe several algorithms to solve the problems in (3) and (4). The basic framework will be to alternate betweens updates to $Z$, $W$, and, if desired, $S$. First we discuss methods for solving for $Z$ with $W$ and $S$ fixed.

### 2.1   Inferring $Z$ from $W$, $X$, and $S$.

The $Z$ update is the most time sensitive, in the sense that the other variables are fixed after training, and only $Z$ is inferred at test time. In general, any iterative algorithm that can be used for the weighted basis pursuit problem can be adapted to our setting; the weights just change at each iteration. We will describe versions of FISTA (Beck and Teboulle, 2009) and coordinate descent (Wu and Lange, 2008; Li and Osher, 2009). While we cannot prove that the algorithms converge to the minimum, in all the applications we have tried, they perform very well.

#### 2.1.1   A FISTA like algorithm

The ISTA (Iterated Shrinkage Thresholding Algorithm) minimizes the energy $||Wz - x||^2 + \lambda|z|_1$ by following gradient steps in the first term with a "shrinkage"; this can be thought of as gradient steps where any coordinate which crosses zero is thresholded. In equations:

$$z^{t+1} = sh_{(\lambda/L)}(Z - \frac{1}{L}W^T(Wz^t - x)),$$

where $sh_a(b) = \text{sign}(b) \cdot h_a(|b|)$, and $h_a(b) = \max(b - a, 0)$. In the case where $z$ is constrained to be nonnegative, $sh$ reduces to $h$. In this paper, $\lambda$ is a vector depending on the current value of $z$, rather than a fixed scalar. After each update, $\lambda$ is updated by $\lambda = \lambda^{t+1} \leftarrow Sz^{t+1}$.

Nesterov's accelerated gradient descent has been found to be effective in the basis pursuit setting, where it is called FISTA (Beck and Teboulle, 2009). In essence one adds to the $z$ update a momentum that approaches one with appropriate speed. Specifically the update equation on $z$

| **Algorithm 1** ISTA | **Algorithm 2** Coordinate Descent |
|---|---|
| **function ISTA**$(X, Z, W, L)$ | **function CoD**$(X, Z, W, S, \bar{S})$ |
|   **Require:** $L >$ largest eigenvalue of $W^T W$. |   **Require:** $\bar{S} = I - W_d^T W_d$ |
|   **Initialize:** $Z = 0$, |   **Initialize:** $Z = 0$; $B = W^T X$; $\lambda = 0$ |
|   **repeat** |   **repeat** |
|     $\lambda = S|z|$ |     $\bar{Z} = h_\lambda(B)$ |
|     $Z = sh_{(\lambda/L)}(Z - \frac{1}{L}W^T(WZ - X))$ |     $k = \operatorname{argmax}|Z - \bar{Z}|$ |
|   **until** change in $Z$ below a threshold |     $B = B + \bar{S}_{.k}(\bar{Z}_k - Z_k)$ |
| **end function** |     $\lambda = \lambda + S_{.k}(\bar{Z}_k - Z_k)$ |
| |     $Z_k = \bar{Z}_k$ |
| |   **until** change in $Z$ is below a threshold |
| |   $Z = h_\alpha(B)$ |
| | **end function** |

becomes $z^{t+1} = y^t + r_t(y^t - y^{t-1})$, $y^t = sh_{(\lambda/L)}(Z - \frac{1}{L}W^T(Wz^t - x))$, $r_t = \frac{u_t - 1}{u_{t+1}}$ and $u_{t+1} = (1 + \sqrt{1 + 4u^2})/2$, $u_1 = 1$. Although our problem is not convex and we do not have any of the normal guarantees, empirically, the Nesterov acceleration works extremely well.

### 2.1.2 Coordinate descent

The coordinate descent algorithm iteratively selects a single coordinate $j$ of $z$, and fixing the other coordinates, does a line search to find the value of $z(k)$ with the lowest energy. The coordinate selection can be done by picking the entry with the largest gradient Wu and Lange (2008), or by approximating the value of the energy after the line search Li and Osher (2009). Suppose at the $t$th step we have chosen to update the $k$th coordinate of $z^t$. Because $S$ is zero on its main diagonal, the penalty term is not quadratic in $z^{t+1}(k)$, but is simply a $\lambda(k)z^{t+1}(k)$, where $\lambda = Sz_k$ (which only depends on the currently fixed coordinates). Thus there is an explicit solution $z^{t+1} = h_\lambda(B(k))$, where $B$ is $W^T(Wz^t - x)$. Just like in the setting of basis pursuit this has the nice property that by updating $B$ and $\lambda$, and using a precomputed $W^T W$, each update only requires $O(K)$ operations, where $K$ is the number of atoms in the dictionary; and in particular the dictionary only needs to be multiplied by $x$ once. In fact, when the actual solution is very sparse and the dictionary is large, the cost of all the iterations is often less than the cost of multiplying $W^T x$.

We will use coordinate descent for a bilinear model below; in this case, we alternate updates of the left coefficients with updates of the right coefficients.

### 2.2 Updating $W$ and $S$

The updates to $W$ and $S$ can be made after each new $z$ is coded, or can be made in batches, say after a pass through the data. In the case of per datapoint updates, we can proceed via a gradient descent: the derivative of all of our models with respect to $W$ for a fixed $x$ and $z$ is $(Wz - x)z^T$. The batch updates to $W$ can be done as in $K$-SVD (Aharon et al., 2006).

It is easier to update $S$ in (4) in batch mode, because of the constraints. With $W$ and $Z$ fixed, the constrained minimization of $S$ is a linear program. We have found that it is useful to average the current $S$ with the minimum of the linear program in the update.

## 3 Experiments

In this section we test the models (3,4) in various experimental settings.

### 3.1 Inference

First we test the speed of convergence and the quality of the resulting state of ISTA, FISTA and coordinate descent algorithms. We use the example of section 3.4 where the input consist of image patches and the connections in $S$ define a tree. The figure 3.1 shows the energy after each iteration

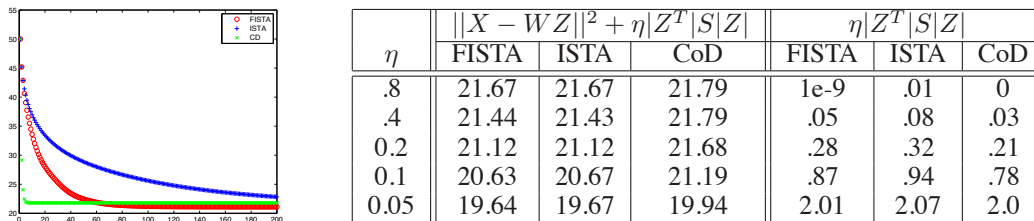

| $\eta$ | $\|X - WZ\|^2 + \eta|Z^T|S|Z|$ | | | $\eta|Z^T|S|Z|$ | | |
|---|---|---|---|---|---|---|
| | FISTA | ISTA | CoD | FISTA | ISTA | CoD |
| .8 | 21.67 | 21.67 | 21.79 | 1e-9 | .01 | 0 |
| .4 | 21.44 | 21.43 | 21.79 | .05 | .08 | .03 |
| 0.2 | 21.12 | 21.12 | 21.68 | .28 | .32 | .21 |
| 0.1 | 20.63 | 20.67 | 21.19 | .87 | .94 | .78 |
| 0.05 | 19.64 | 19.67 | 19.94 | 2.01 | 2.07 | 2.0 |

Figure 2: On the left: The energy values after each iteration of the 3 methods, averaged over all the data points. On the right: values of the average energy $\frac{1}{N}\sum_j \|WZ_j - X_j\|^2 + \eta|Z_j|^T S|Z_j|$ and average S sparsity $\frac{1}{N}\sum_j |Z_j|^T S|Z_j|$. The "oracle" best tree structured output computed by using an exhaustive search over the projections of each data point onto each branch of the tree has the average energy 20.58 and sparsity 0. $S, W$, and $X$ are as in section 3.4

of the three methods average over all data points. We can see that coordinate descent very quickly moves to its resting state (note that each iteration is much cheaper as well, only requiring a few column operations), but does not on average tend to be quite as good a code as ISTA or FISTA. We also see that FISTA gets as good a code as ISTA but after far fewer iterations.

To test the absolute quality of the methods, we also measure against the "oracle" - the lowest possible energy when none of the constraints are broken, that is, when $|z|S|z| = 0$. This energy is obtained by exhaustive search over the projections of each data point onto each branch of the tree. In the table in Figure 3.1, we give the values for the average energy $\frac{1}{N}\sum_j \|WZ_j - X_j\|^2 + \eta|Z_j|^T S|Z_j|$ and for the sparsity energy $\frac{1}{N}\sum_j \eta|Z_j|^T S|Z_j|$ for various values of $\eta$. Notice that for low values of $\eta$, the methods presented here give codes with better energy than the best possible code on the tree, because the penalty is small enough to allow deviations from the tree structure; but when the $\eta$ parameter is increased, the algorithms still compare well against the exhaustive search.

## 3.2 Scaling

An interesting property of the models (3,4) is their scaling: if the input is re-scaled by a constant factor the optimal code is re-scaled by the same factor. Thus the model preserves the scale information and the input doesn't need to be normalized. This is not the case in the standard $l_1$ sparse coding model (1). For example if the input becomes small the optimal code is zero.

In this subsection we train the model (3) on image patches. In the first part of the experiment we preprocess each image patch by subtracting its mean and set the elements of $S$ to be all equal and positive except for zeros on the diagonal. In the second part of the experiment we use the original image patches without any preprocessing. However since the mean is the strongest component we introduce the first example of structure: We select one of the components of $z$ and disconnect it from all the other components. The resulting $S$ is equal to a positive constant everywhere except on the diagonal, the first row, and the first column, where it is zero. After training in this setting we obtain the usual edge detectors (see Figure (3a)) except for the first component which learns the mean. In the first setting the result is simply a set of edge detectors. Experimentally, explicitly removing the mean before training is better as the training converges a lot more quickly.

## 3.3 Locally connected net

In this section we impose a structure motivated by the connectivity of cortical layer V1. The cortical layer has a two dimensional structure (with depth) with locations corresponding to the locations in the input image. The sublayer 4 contains simple cells with edge like receptive fields. Each such cell receives input from a small neighborhood of the input image at its corresponding location. We model this by placing units in a two dimensional grid above the image and connecting each unit to a small neighborhood of the input, Figure 1. We also bind connections weights for units that are far enough from each other to reduce the number of parameters without affecting the local structure (Gregor and LeCun, 2010). Next we connect each unit by inhibitory interactions (the $S$ matrix) to units in its ring-shaped neighborhood: there is a connection between two units if their distance $d$

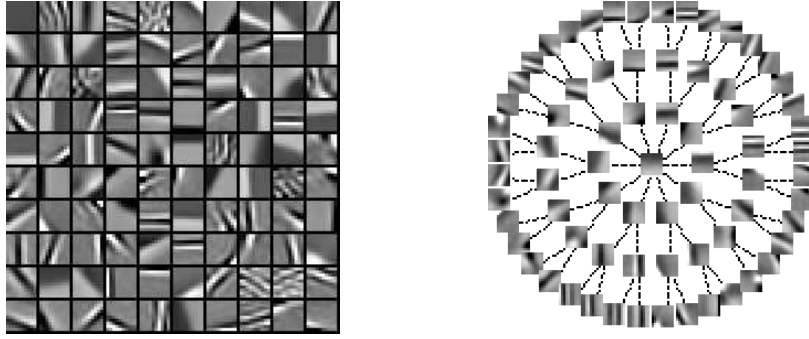

Figure 3: (a) Filters learned on the original unprocessed image patches. The $S$ matrix was fully connected except the unit corresponding to the upper left corner which was not connected to any other unit and learned the mean. The other units typically learned edge detectors. (b) Filters learned in the tree structure. The $S_{ij} = 0$ if one of the $i$ and $j$ is descendant of the other and $S_{ij} = S^0 d(i,j)$ otherwise where $d(i,j)$ is the distance between the units in the tree. The filters in a given branch are of a similar orientation and get refined as we walk down the tree.

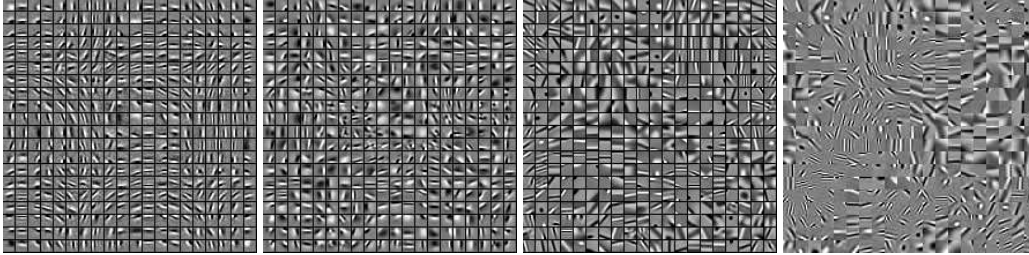

Figure 4: (a-b) Filters learned on images in the locally connected framework with local inhibition shown in the Figure 1. The local inhibition matrix has positive value $S_{ij} = S^0 > 0$ if the distance between code units $Z_i$ and $Z_j$ satisfies $r_1 < d(i,j) < r_2$ and $S_{ij} = 0$ otherwise. The input size was $40 \times 40$ pixels and the receptive field size was $10 \times 10$ pixels. The net learned to place filters of similar orientations close together. (a) Images were preprocessed by subtracting the local mean and dividing by the standard deviation, each of width 1.65 pixels. The resulting filters are sharp edge detectors and can therefore be naturally imbedded in two dimensions. (b) Only the local mean, of width 5 pixels, was subtracted. This results in a larger range of frequencies that is harder to imbed in two dimensions. (c-d) Filters trained on $10 \times 10$ image patches with mean subtracted and then normalized. (c) The inhibition matrix was the same as in (a-b). (d) This time there was an $l_1$ penalty on each code unit and the lateral interaction matrix $S$ was excitatory: $S_{ij} < 0$ if $d(i,j) < r_2$ and zero otherwise.

satisfies $r_1 < d < r_2$ for some radii $r_1$ and $r_2$ (alternatively we can put $r_1 = 0$ and create excitatory interactions in a smaller neighborhood). With this arrangement units that turn on simultaneously are typically either close to each other (within $r_1$) or far from each other (more distant than $r_2$).

Training on image patches results in the filters shown in the Figure figure 4. We see that filters with similar orientations are placed together as is observed in V1 (and other experiments on group sparsity, for example (Hyvarinen and Hoyer, 2001)). Here we obtain these patterns by the presence of inhibitory connections.

## 3.4 Tree structure

In this experiment we place the units $z$ on a tree and desire that the units that are on for a given input lie on a single branch of the tree. We define $S_{ij} = 0$ if $i$ is descendant of $j$ or vice versa and

$S_{ij} = S^0 d(i,j)$ otherwise where $S^0 > 0$ is a constant and $d(i,j)$ is the distance between the nodes $i$ and $j$ (the number of links it takes to get from one to the other).

We trained (3) on image patches. The model learns to place low frequency filters close to the root of the tree and as we go down the branches the filters "refine" their parents, Figure 3b.

## 3.5   A convolutional image parts model

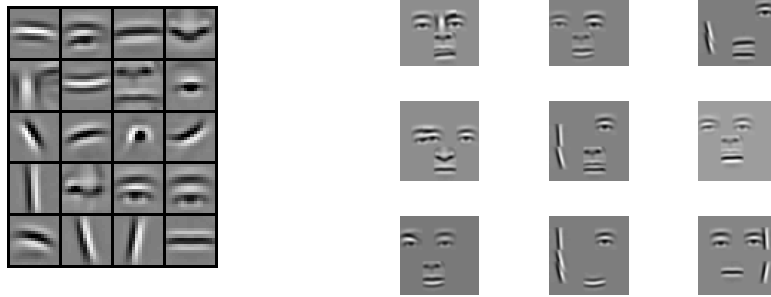

Figure 5: On the left: the dictionary of $16 \times 16$ filters learned by the convolutional model on faces. On the right: some low energy configurations, generated randomly as in Section 3.5 . Each active filter has response 1.

We give an example of learning $S$ in a convolutional setting. We use the centered faces from the faces in the wild dataset, available at `http://vis-www.cs.umass.edu/lfw/`. From each of the 13233 images we subsample by a factor of two and pick a random $48 \times 48$ patch. The $48 \times 48$ image $x$ is then contrast normalized to $x - b * x$, where $b$ is a $5 \times 5$ averaging box filter; the images are collected into the $48 \times 48 \times 13233$ data set $X$.

We then train a model minimizing the energy

$$\sum_i ||\sum_{j=1}^{20} W_j * z_{ji} - X_i||^2 + p(z)^T S p(z),$$

$$\beta \geq S \geq 0, \ S = S^T, \ |S_j|_1 = \alpha.$$

Here the code vector $z$ is written as a $48 \times 48 \times 20$ feature map. The pooling operator $p$ takes the average of the absolute value of each $8 \times 8$ patch on each of the 20 maps, and outputs a vector of size $6 \cdot 6 \cdot 20 = 720$. $\beta$ is set to 72, and $\alpha$ to .105. Note that these two numbers roughly specify the number of zeros in the solution of the $S$ problem to be 1600.

The energy is minimized via the batch procedure. The updates for $Z$ are done via coordinate descent (coordinate descent in the convolutional setting works exactly as before), the updates for $W$ via least squares, and at each update, $S$ is averaged with .05 of the solution to the linear program in $S$ with fixed $Z$ and $W$. $W$ is initialized via random patches from $X$, and $S$ is initialized as the all ones matrix, with zeros on the diagonal. In Figure 5 the dictionary $W$ is displayed.

To visualize the $S$ which is learned, we will try to use it to generate new images. Without any data to reconstruct the model will collapse to zero, so we will constrain $z$ to have a fixed number of unit entries, and run a few steps of a greedy search to decide which entries should be on. That is: we initialize $z$ to have 5 random entries set to one, and the rest zero. At each step, we pick one of the nonzero entries, set it to zero, and find the new entry of $z$ which is cheapest to set to one, namely, the minimum of the entries in $Sp(z)$ which are not currently turned on. We repeat this until the configuration is stable. Some results are displayed in 5.

The interesting thing about this experiment is the fact that no filter ever is allowed to see global information, except through $S$. However, even though $W$ is blind to anything larger than a $16 \times 16$ patch, through the inhibition of $S$, the model is able to learn the placement of facial structures and long edges.

# References

Ackley, D., Hinton, G., and Sejnowski, T. (1985). A learning algorithm for boltzmann machines*. *Cognitive science*, 9(1):147–169.

Aharon, M., Elad, M., and Bruckstein, A. (2006). K-SVD: An algorithm for designing overcomplete dictionaries for sparse representation. *IEEE Transactions on Signal Processing*, 54(11):4311–4322.

Baraniuk, R. G., Cevher, V., Duarte, M. F., and Hegde, C. (2009). Model-Based Compressive Sensing.

Beck, A. and Teboulle, M. (2009). A fast iterative shrinkage-thresholding algorithm with application to wavelet-based image deblurring. *ICASSP'09*, pages 693–696.

Druckmann, S. and Chklovskii, D. (2010). Over-complete representations on recurrent neural networks can support persistent percepts.

Garrigues, P. and Olshausen, B. (2008). Learning horizontal connections in a sparse coding model of natural images. *Advances in Neural Information Processing Systems*, 20:505–512.

Garrigues, P. and Olshausen, B. (2010). Group sparse coding with a laplacian scale mixture prior. In Lafferty, J., Williams, C. K. I., Shawe-Taylor, J., Zemel, R., and Culotta, A., editors, *Advances in Neural Information Processing Systems 23*, pages 676–684.

Gregor, K. and LeCun, Y. (2010). Emergence of Complex-Like Cells in a Temporal Product Network with Local Receptive Fields. *Arxiv preprint arXiv:1006.0448*.

Hopfield, J. (1982). Neural networks and physical systems with emergent collective computational abilities. *Proceedings of the National Academy of Sciences of the United States of America*, 79(8):2554.

Huang, J., Zhang, T., and Metaxas, D. N. (2009). Learning with structured sparsity. In *ICML*, page 53.

Hyvarinen, A. and Hoyer, P. (2001). A two-layer sparse coding model learns simple and complex cell receptive fields and topography from natural images. *Vision Research*, 41(18):2413–2423.

Jacob, L., Obozinski, G., and Vert, J.-P. (2009). Group lasso with overlap and graph lasso. In *Proceedings of the 26th Annual International Conference on Machine Learning*, ICML '09, pages 433–440, New York, NY, USA. ACM.

Jenatton, R., Mairal, J., Obozinski, G., and Bach, F. (2010). Proximal methods for sparse hierarchical dictionary learning. In *International Conference on Machine Learning (ICML)*.

Kavukcuoglu, K., Ranzato, M., Fergus, R., and LeCun, Y. (2009). Learning invariant features through topographic filter maps. In *Proc. International Conference on Computer Vision and Pattern Recognition (CVPR'09)*. IEEE.

Kim, S. and Xing, E. P. (2010). Tree-guided group lasso for multi-task regression with structured sparsity. In *ICML*, pages 543–550.

Li, Y. and Osher, S. (2009). Coordinate descent optimization for l1 minimization with application to compressed sensing; a greedy algorithm. *Inverse Problems and Imaging*, 3(3):487–503.

Olshausen, B. and Field, D. (1996). Emergence of simple-cell receptive field properties by learning a sparse code for natural images. *Nature*, 381(6583):607–609.

Wu, T. T. and Lange, K. (2008). Coordinate descent algorithms for lasso penalized regression. *ANNALS OF APPLIED STATISTICS*, 2:224.

